# Searching for objects driven by context

**Bogdan Alexe**
BIWI
ETH Zurich

**Nicolas Heess**
Gatsby Unit
UCL

**Yee Whye Teh**
Department of Statistics
University of Oxford

**Vittorio Ferrari**
School of Informatics
University of Edinburgh

## Abstract

The dominant visual search paradigm for object class detection is sliding windows. Although simple and effective, it is also wasteful, unnatural and rigidly hardwired. We propose strategies to search for objects which intelligently explore the space of windows by making sequential observations at locations decided based on previous observations. Our strategies adapt to the class being searched and to the content of a particular test image, exploiting context as the statistical relation between the appearance of a window and its location relative to the object, as observed in the training set. In addition to being more elegant than sliding windows, we demonstrate experimentally on the PASCAL VOC 2010 dataset that our strategies evaluate two orders of magnitude fewer windows while achieving higher object detection performance.

## 1   Introduction

Object class detection is a central problem in computer vision. Among the broad palette of approaches [2, 22, 31], most state-of-the-art detectors rely on the sliding window paradigm [7, 8, 12, 15, 30, 31]. A classifier is trained to decide whether a window contains an instance of the target class and is used at test time to score *all* windows in an image over a regular grid in location and scale. The local maxima of the score function are returned as the detections. Despite its popularity, the sliding window paradigm seems wasteful and unnatural. Cognitive science research [24] measuring eye-tracks has shown that humans search for objects in a very different way, by successively exploring a *small* number of promising locations, rapidly converging on the object of interest. This process decides where to look next based on the context gathered in previous observations (fixation points). As opposed to sliding-windows, this search scheme adapts to the image content and the class being searched.

In this paper we propose strategies to search for objects in images which have these crucial characteristics. Each strategy is specific to an object class and intelligently explores the space of windows by making sequential observations at locations decided based on previous observations. Figure 1 illustrates the key intuition by applying an ideal strategy to search for cars in a test image. The strategy might start at window $w_1$, which is a patch of sky. The strategy has learned from the training data that cars are typically below the sky, so it decides to try a window below $w_1$, e.g. moving to window $w_2$. As $w_2$ covers a patch of road, and the strategy has learned that cars are frequently found on roads, it continues to search the road region, e.g. moving to $w_3$. As $w_3$ contains the right end of a car, the strategy moves to the left to $w_4$, completing the search.

Given a set of training images of a class with ground-truth object locations, our method learns a strategy to localize objects of that class by sequentially evaluating windows. To achieve this it models the statistical relation between the position and appearance of windows in the training images to their relative position wrt to the ground-truth (sec. 2 and 3). In addition to being more elegant than sliding windows, the proposed technique offers practical advantages. It greatly reduces the number of observed windows, and therefore the number of times a window classifier is evaluated (potentially very expensive [15, 30]). Moreover, it naturally exploits context information to avoid evaluating the classifier on large portions of an image which might contain cluttered areas. This leads to lower

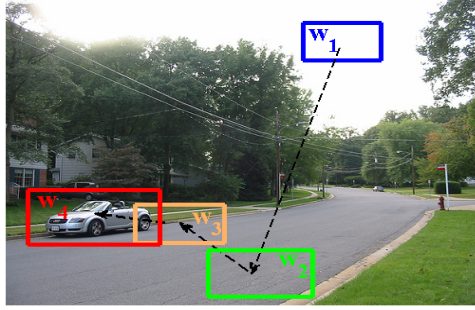

Figure 1: **Searching for a car driven by context.** *An ideal search strategy moves through the sequence of windows $w_1$ to $w_4$. See main text.*

false-positive rates, and therefore *higher* object detection performance than sliding windows, despite evaluating fewer windows. Finally, our method makes no assumption on the form of the window classifier and therefore can be applied on top of *any* classifier (e.g. [7, 8, 12, 15, 30, 31]).

In sec. 5 we report experiments on the highly challenging PASCAL VOC 2010 dataset, using the popular deformable part model of [12] as the window classifier. The experiments demonstrate that our learned strategies perform better in terms of object detection accuracy than sliding windows, while greatly reducing the number of classifier evaluations by a factor of $250\times$ (100 vs 25000 in [12]). Moreover, we outperform two recent methods to reduce the number of classifier evaluations [1, 29] as they evaluate about 1000 windows while losing detection accuracy compared to sliding windows. To our knowledge, this is the first method capable of saving window evaluations while at the same time improving detection accuracy.

**Related work.**   Several works try to reduce the number of windows evaluated in the traditional sliding-window paradigm. Lampert et al. [20] proposed a branch-and-bound scheme to find the highest scored window while evaluating the classifier as few times as possible. However, it is restricted to classifiers for which a good upper bound on a set of windows exists. Other works [15, 30] run first a linear classifier over all windows, and then evaluate a complex non-linear kernel only on a few highly scored windows. The recent approaches [1, 29] evaluate the classifier only on a small number of windows likely to cover objects rather than backgrounds. The authors of [11, 26] propose a complementary tactic: to reduce the cost of evaluating one window, but stay in the sliding-window paradigm. Their techniques are specific to the window classifier [12], as they exploit its exact form (e.g. parts [11], two resolutions [26]).

Context has been used by [6, 8, 16, 28] to improve object detection. They employ background-to-object context to avoid out-of-context false-positive detections [16, 28], or reason about the spatial relations between multiple objects [6, 8]. All these methods use context as an additional cue on top of individual object detectors, whereas in our approach context drives the search for an object in the image, determining the sequence of windows where the classifier is evaluated.

Numerous works propose saliency detectors [1, 13, 17, 18] which try to find interesting regions in an image corresponding to objects of *any* class. These are often inspired by human eye movements [9, 19]. Our goal instead is to devise a search strategy specific to one particular class, that can exploit the relation between context appearance and the position of instances of that class in training images.

Closest to our work are techniques that consider vision with sequential fixations as a task-oriented learning problem [4, 5, 21, 25]. Analog to our work, [5] reduces the number of window classifier evaluations, avoiding the wasteful sliding window scheme. However, it only considers the output of the window classifier and therefore cannot exploit context. Our search instead is driven by the relation between the appearance of a window and the relative location of the object, as learned from annotated training images. This has the added benefit of improving object detection accuracy compared to sliding windows. Importantly, to our knowledge, no previous approach has been demonstrated on a dataset of difficulty similar to PASCAL VOC.

The rest of the paper is organized as follows. Sec. 2 gives an overview of our new method to localize objects, followed by a detailed presentation in sec. 3. In sec. 4 we discuss the most important implementation issues and conclude with experiments in sec. 5.

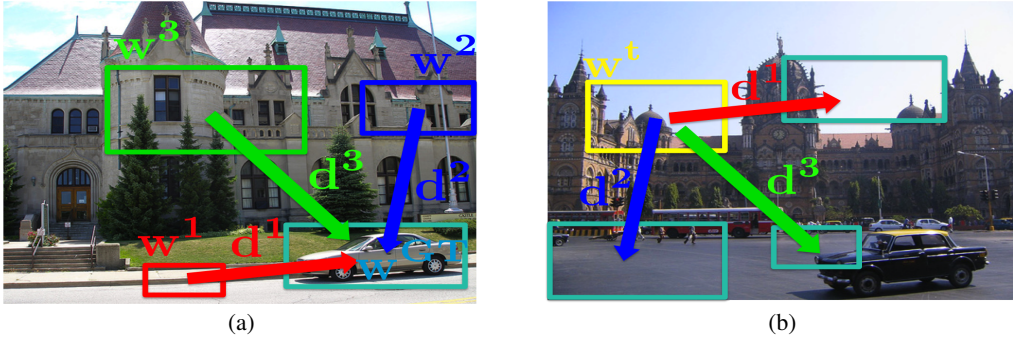

<div align="center">(a)                              (b)</div>

Figure 2: **Displacement vector.** *(a) Three windows $\mathbf{w}^l$ in a training image and their displacement vector $\mathbf{d}^l$. (b) A test image. Applying $\mathbf{d}^l$ to the current observation window $\mathbf{w}^t$ results in the translated windows $\mathbf{w}^t \oplus \mathbf{d}^l$.*

## 2 Overview of our method

Our method detects an object in a test image with a sequential process, by evaluating one window $\mathbf{y}^t$ at each time step $t$. Over time, it gradually integrates these local observations into a global estimate of the object location in the image. At each time step, it actively decides which window to evaluate next based on all past observations, trying to acquire observations that will improve the global location estimate. This decision process is learned from a set of images labeled with ground-truth bounding-boxes on all instances of the object class. The key driving force here is the statistical dependency between the position/appearance of a window and the ground-truth location of the object (e.g. cars are often on roads; boats are often below sky). Our method first finds training windows similar in position/appearance to the current window $\mathbf{y}^t$ in the test image. Then, each such training window votes for a possible object location in the test image through its displacement vector relative to the ground-truth object (fig. 2). At each time step these votes are accumulated into a probabilistic map of possible object locations (fig. 3). The maps are then integrated over time and used to decide which window to evaluate next (sec. 3.1).

The behavior of our decision process is controlled by the weights of the various features in the similarity measure used to compare windows in the test image to training windows. We adapt these weights to each class by optimizing the accuracy by which the strategy localize training object instances in a single time step (sec. 3.3).

The process involves comparing high-dimensional appearance descriptors between a test window $\mathbf{y}^t$ and hundreds of thousand training windows. We greatly reduce the cost of these comparisons by embedding the descriptors in a lower-dimensional Hamming space using [14] (sec. 4).

## 3 Context-driven search

In this section we describe our method in detail. Given a test image $\mathbf{x}$, it sequentially collects a fixed number $T$ of observations $\mathbf{y}^t$ for windows $\mathbf{w}^t$ before making a final detection decision. At each time step $t$ the next observation window is chosen based on all past observations. Thus, we try to solve two tightly connected problems:

(1) At each time step $t < T$, given past observations $\mathbf{y}^1 \dots \mathbf{y}^t$ obtained for windows $\mathbf{w}^1 \dots \mathbf{w}^t$ we need to actively choose the window $\mathbf{w}^{t+1}$ where to make the next observation $\mathbf{y}^{t+1}$. In section 3.1 we formalize this in terms of a mapping $\pi^S$ from past observations to the next observation window. We refer to this mapping as our *search policy* (sec. 3.1).

(2) At the last time step $t = T$, given all observations $\mathbf{y}^1 \dots \mathbf{y}^T, \mathbf{w}^1 \dots \mathbf{w}^T$, we need to make a final decision about the object location. We refer to this mapping as our *output policy* $\pi^O$ (sec. 3.2).

The two problems are tied since the observations made at time steps $1 \dots T$ affect our ability to detect the object. Hence, we want to pick a search policy $\pi^S$ that chooses windows leading to observations that enable the output policy $\pi^O$ to make a good detection decision. In sec. 3.1 and 3.2 we explain how to tackle these problems individually. We then discuss how the parameters of the search policy can be adapted to a particular class to optimize detection accuracy (sec. 3.3).

<div align="center">3</div>

In the following we assume that a window $\mathbf{w}^t = (x^t, y^t, s^t)$ is defined by its $x, y$ location and scale $s$. In any given image $\mathbf{x}$ there is a fixed set of windows from a dense grid in $x$, $y$ and scale space that depends on the image size and the aspect ratio of the class under consideration (see sec. 4). An observation consists of $J$ feature vectors $\mathbf{f}_j^t$ which describe a window $\mathbf{y}^t = (\mathbf{f}_1^t, \ldots, \mathbf{f}_J^t)$. Sec. 4 details the specific grid and window features we use.

## 3.1 Search policy

The search policy $\pi^S$ determines the choice of the next observation window given the observation history at time step $t$. We formalize this in terms of a mapping $\mathbf{w}^{t+1} = \pi^S(\mathbf{w}^1, \mathbf{y}^1, \ldots, \mathbf{w}^t, \mathbf{y}^t)$ from past observations to the next observation window. This mapping is based on a conditional distribution $M^t(\mathbf{w} | \mathbf{w}^1, \mathbf{y}^1, \ldots, \mathbf{w}^t, \mathbf{y}^t; \Theta)$ over all possible candidate observations locations $\mathbf{w}$ in the test image given the past observation windows. $\Theta$ are the parameters of $M$. The mapping chooses the window with highest probability in $M$ as the next observation window

$$\mathbf{w}^{t+1} = \pi^S(\mathbf{w}^1, \mathbf{y}^1, \ldots, \mathbf{w}^t, \mathbf{y}^t) = \arg\max_{\mathbf{w}} M^t(\mathbf{w} | \mathbf{w}^1, \mathbf{y}^1, \ldots, \mathbf{w}^t, \mathbf{y}^t; \Theta). \qquad (1)$$

The conditional distribution $M^t$ is parameterized in terms of a set of probabilistic *vote maps* $m(\mathbf{w} | \mathbf{w}^t, \mathbf{y}^t)$. These maps are obtained independently at each time step and can be seen as distributions over windows $\mathbf{w}$, given the information about the image from that time step only. In the following we explain how the individual vote maps are obtained at a time step $t$, and describe then how they are integrated over time to form the full conditional distribution $M^t$.

**Vote maps.** Individual vote maps are represented in a non-parametric manner which enables us to capture complex dependencies between observations and object locations. As usual in object detection, our method is given for training a set of images labeled with ground-truth bounding-boxes on object instances of the class. We sample a large number $L$ of windows $\mathbf{w}^l$ uniformly from all training images, and we store their position $\mathbf{w}^l = (x^l, y^l, s^l)$, the associated feature vectors $\mathbf{y}^l$, as well as the displacement vectors $\mathbf{d}^l$ that record the location of the ground-truth object relative to a window. Each window in a training image can use this to vote for the relative position $\mathbf{d}^l$ where it expects the object to be. Given the current observation $\mathbf{y}^t$ for image window $\mathbf{w}^t$ in the test image, the distribution over object positions is then given by the spatial distribution of these votes

$$\tilde{m}(\mathbf{w}; \mathbf{w}^t, \mathbf{y}^t, \Theta) = \sum_{l=1}^{L} K_F(\mathbf{y}^t, \mathbf{y}^l; \Theta^F) \cdot K_S(\mathbf{w}, \mathbf{w}^t \oplus \mathbf{d}^l; \Theta^S). \qquad (2)$$

Here, $K_F$ is a kernel that measures the similarity between the features describing two windows and is used to weight each vote; $K_S$ is a spatial smoothing kernel; $\Theta^S$, $\Theta^F$ are the kernel parameters the operator $\oplus$ translates a window $\mathbf{w}^t$ by the displacement vector $\mathbf{d}^l$ (after appropriately rescaling it to compensate for the potentially different size of the training and test images).

The summation over all $L$ training windows is computationally expensive. In practice we truncate it and consider only the $Z$ training windows most similar to the current observation window $\mathbf{y}^t$. Hence, eq. (2) can be seen as a soft nearest-neighbor estimator (NN, fig. 3). [1]

Different choices for $K_F$ and $K_S$ are conceivable. For $K_F$ we use an exponential function on distances computed separately for each type $j$ of feature vector $\mathbf{f}_j$ describing a window (see sec. 4)

$$K_F(\mathbf{y}^t, \mathbf{y}^l; \Theta^F) = \exp\left\{ -\sum_{j=1}^{J} \theta_j^F d_j(\mathbf{f}_j^t, \mathbf{f}_j^l) \right\}, \qquad (3)$$

with $d_j(\cdot, \cdot)$ being the distance between two feature vectors of type $j$. The scalar parameters $\theta_j^F$ weight the contributions of the various feature types. For $K_S$ we use the area of intersection divided by the area of union [10]. This choice of $K_S$ has no parameters $\Theta^S$, but forms with free parameters are also possible, e.g. a Gaussian kernel.

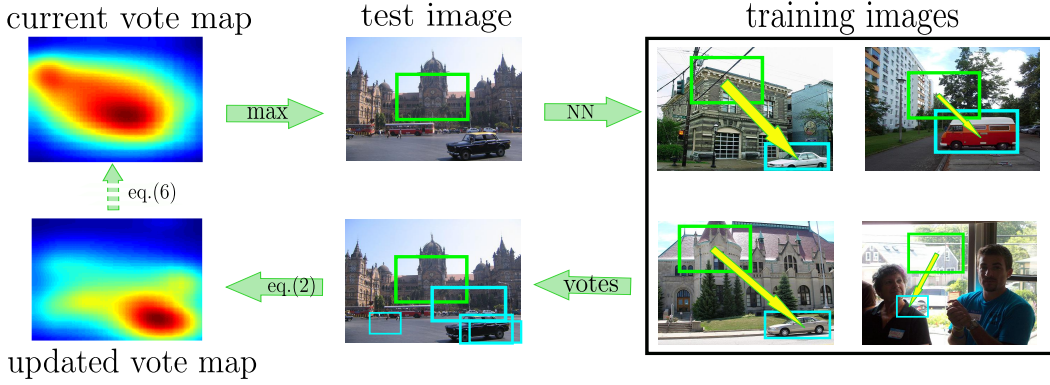

Figure 3: **Search policy during one time step.** *The next observation window (green) is chosen as the highest probability window in the current vote map $M^t$. Next we retrieve the $Z$ most similar training windows according to $K_F$ (NN arrow, for 'nearest neighbors'). Each NN votes through its displacement vector (yellow arrow) pointing to the ground-truth object (cyan). This leads to the vote map $m$ specific to this time step $t+1$ (eq. (2) and (4)), which is finally integrated into the updated whole map $M^{t+1}$ (eq. (6)). The cycle then repeats in $t+2$.*

**Integrating vote maps over time.**   Normalizing the vote map $\tilde{m}(\mathbf{w}; \mathbf{w}^t, \mathbf{y}^t, \Theta^t)$ in eq. (2) at a time step $t$ yields a conditional distribution over candidate observation locations given the observation $\mathbf{y}^t$ at window $\mathbf{w}^t$

$$m(\mathbf{w}|\mathbf{w}^t, \mathbf{y}^t, \Theta) = \frac{\tilde{m}(\mathbf{w}; \mathbf{w}^t, \mathbf{y}^t, \Theta)}{\sum_{\mathbf{w}'} \tilde{m}(\mathbf{w}'; \mathbf{w}^t, \mathbf{y}^t, \Theta)} \tag{4}$$

In order to obtain $M^t(\mathbf{w}|\mathbf{w}^1 \mathbf{y}^1 \dots \mathbf{w}^t \mathbf{y}^t; \Theta)$ we integrate the normalized vote maps over all past time steps $1 \dots t$ using an exponentially decaying mixture

$$M^t(\mathbf{w}|\mathbf{w}^1, \mathbf{y}^1, \dots, \mathbf{w}^t, \mathbf{y}^t; \Theta) = \sum_{t'=1}^{t} a(t, t') m(\mathbf{w}|\mathbf{w}^{t'}, \mathbf{y}^{t'}, \Theta), \tag{5}$$

where $a(t, t') = \alpha(1 - \alpha)^{t-t'}$ for $t' > 1$ and $a(1, t) = (1 - \alpha)^{t-1}$ for some constant $0 < \alpha \le 1$.[2]

Defining $M^t$ as in eq. (5) has two appealing interpretations. Firstly, we can see the full vote map $M^t$ at time step $t$ as a sufficient statistic of past observations which is updated recursively

$$M^{t+1}(\mathbf{w}|(\mathbf{w}^1, \mathbf{y}^1, \dots, \mathbf{w}^{t+1}, \mathbf{y}^{t+1}) = \alpha m(\mathbf{w}|\mathbf{w}^{t+1}, \mathbf{y}^{t+1}) + (1 - \alpha)M^t(\mathbf{w}|\mathbf{w}^1, \mathbf{y}^1, \dots, \mathbf{w}^t, \mathbf{y}^t). \tag{6}$$

so that we only need to store the latest full vote map. Even though the next observation window is chosen deterministically (eq. (1)), by deriving it from the probabilistic vote-map $M^t$ and updating this map over time we are effectively maintaining an estimate of the uncertainty about which are good candidate windows to visit in the next step. Secondly, $M^t$ should not be seen as a posterior distribution over actual object locations. It should be seen as a policy to propose windows that should be visited in the future. Each past observation independently proposes candidate observation locations which are later visited if they accumulate enough support over time. The exponential decay ensures that the influence of observations made a long time ago gets gradually forgotten and therefore encourages exploration. This makes particular sense in combination with the output policy that we employ (see sec. 3.2) for which it is sufficient to have visited the correct object location once over the course of the whole search history.

### 3.2   Output policy

After obtaining $T$ observations $\mathbf{y}^1 \dots \mathbf{y}^T$ for $T$ different windows $\mathbf{w}^1 \dots \mathbf{w}^T$ in the test image, our strategy must output a single window which it believes to be the most likely to contain an object of the class of interest. As our strategy is designed to visit good candidate windows over the course

of its search, we simply output as the final detection the visited window that has the highest score according to a window classifier $c$ trained beforehand for that class [12] (see sec. 4)

$$\mathbf{w}^{\text{out}} = \arg\max_t c(\mathbf{w}^t) \tag{7}$$

## 3.3 Learning weights

Our search policy involves the feature weights $\Theta^F = \{\theta_j^F\}$ in the window similarity kernel (eq. (3)), which need to be set for each class. Directly maximizing the detection rate after $T$ steps is difficult for several reasons: the detection rate is piecewise constant and non-continuous wrt. the parameters $\Theta^F$; the search is a sequential decision process where window selected at different time steps depend on each other; the policy is non-differentiable with-respect to $\Theta^F$ (due to the max in eq. (1)). We therefore use an approximate learning procedure that iteratively optimizes the *one-step* detection performance of the *stochastic* vote maps $M^t$ (eq. (5)).

Given a training set of images with ground-truth object bounding-boxes, we partition it into two equal-sized disjoint subsets. The first subset provides the $L$ training windows for the non-parametric representation of $m$ in eq. (2). On the second subset we run a stochastic version of our search strategy in which we *sample* the next observation window according to $\mathbf{w}^{t+1} \sim M^t(\cdot|\mathbf{w}^1, \mathbf{y}^1, \ldots, \mathbf{w}^t, \mathbf{y}^t; \Theta)$ (instead of taking the max). Running the strategy once on the $b$-th training image produces a sample sequence of windows and associated observations $\hat{\mathbf{h}} = (\hat{\mathbf{w}}^1, \hat{\mathbf{y}}^1, \ldots, \hat{\mathbf{w}}^T, \hat{\mathbf{y}}^T)$. Given this sequence we then improve the following objective

$$\mathcal{J}(\Theta^F; \hat{\mathbf{h}}) = \sum_{t=1}^{T} \sum_{\mathbf{w}} M^t(\mathbf{w}|\hat{\mathbf{w}}^1, \hat{\mathbf{y}}^1, \ldots, \hat{\mathbf{w}}^t, \hat{\mathbf{y}}^t; \Theta) \cdot K_S(\mathbf{w}, \mathbf{w}_{\text{GT}}^b) \tag{8}$$

by updating the parameters by the gradient of $\mathcal{J}$: $\Theta_{\text{new}}^F = \Theta_{\text{old}}^F + \gamma \nabla_{\Theta^F} \mathcal{J}(\Theta_{\text{old}}^F; \hat{\mathbf{h}})$. We denote with $\mathbf{w}_{\text{GT}}^b$ the ground-truth object location. At the beginning of learning we initialize all $\theta_j^F$ to 1 and then perform several hundred updates as described, cycling through the training images. Each update involves re-running the strategy on a training image to obtain a sample history $\mathbf{h}$.[3]

The objective (8) tries to maximize the overlap $K_S$ with the ground-truth bounding-box weighted by $M^t$, hence encouraging the policy to choose for the next step windows that are likely to lie on the object to be detected. While it leads to good results, our learning procedure is only an approximation. In particular, it optimizes the weights to maximize only the single-step performance. [4]

## 4 Implementation details

**Window classifier.**    As window classifier we choose the popular multiscale deformable part model of [12]. This model includes one root HOG filter [7] plus several part HOG filters with their associated deformation models. The score of a window at location $(x, y, s)$ is a weighted sum of the score of the root filter, the part filters and a deformation cost measuring the deviation of the part from its mean training position. The work [12] also defines a multiscale image grid which forms the fixed set of windows observable by our method (sec. 3). Note how all windows in this grid have the same aspect-ratio, as there is a separate window classifier per object viewpoint [12].

**Window features.**    The kernel $K_F$ used for computing the similarity $K_F$ between two windows in eq. (3) involves $J = 3$ feature types: (j=1) $\mathbf{f}_1$ is the $(x, y, s)$ location normalized by image size; (j=2) $\mathbf{f}_2$ is a histogram of oriented gradients (HOG) [7]; (j=3) $\mathbf{f}_3$ is the score of the window classifier $c$ [12]. Their corresponding distance functions $d_j$ are: (j=1) $d_1 = 1 - K_S(\mathbf{f}_1^t, \mathbf{f}_1^l)$, i.e. the intersection-over-union of the two windows; (j=2) $d_2$ is the normalized Hamming distance between binary string

Table 1: **Object detection on PASCAL VOC10.** *For each method we show the average number of windows evaluated per image (#win), the detection rate (DR) and the mean average precision (mAP) over all 54 class-viewpoint combinations. See main text for details.*

|  | Sliding Window [12] | Our | Random Chance | Objectness [1] | Selective Search [29] |
|---|---|---|---|---|---|
| mAP | 0.266 | **0.293** | 0.070 | 0.259 | 0.261 |
| DR | 0.372 | **0.409** | 0.124 | 0.366 | 0.370 |
| #win | 25000 | **100** | 100 | 1046 | 408 |

representations of the HOG (see below); (iii) $d_3 = |c(\mathbf{w}^t) - c(\mathbf{w}^l)|$ is the absolute difference in the window classifier score. This is a measure of appearance similarity from the viewpoint of the classifier. It encourages the search policy to continue to probe nearby windows after one observation hits an instance of the class.

**Rapid window similarity.** We embed the window appearance features (HOG) in a Hamming space of dimension 128 using [14], thus going from 49600 bits to just 128. This has two advantages. First, it reduces the memory footprint for storing the appearance descriptor for all training windows of a class to the point where they all fit in memory at once. Second, it greatly speeds up the computation of the similarity $K_F$ between two windows, from about 500'000 per second for the original HOG to 65 million per second for the Hamming embedded version (i.e. $130\times$ faster). This speedup is very useful as the number of training windows is typically very large (from a few hundred thousands up to a million depending on the class in our experiments).

## 5 Experiments and conclusions

**Dataset and protocol.** We evaluate the ability of our learned strategies to detect objects on the very popular and highly challenging PASCAL VOC 2010 dataset [10], which contains object instances from 20 classes (e.g. car, sheep, motorbike) annotated by bounding-boxes and viewpoint labels (e.g. left, front). The objects appear in cluttered backgrounds and vary greatly in location, scale, appearance, viewpoint and illumination (fig. 4). The dataset is composed of three subsets train, val and test. We train our method on all class-viewpoint combinations for which at least 20 training images are available, for a total of 54 combinations (each is referred to as a *class* from now on). As bounding-box annotations for the test subset are not available, we use val as the test set. For each class the test set consists of all images that contain an instance of that class (positive images) and an equal number of randomly sampled negative images. We quantify performance by three factors: **(mAP)** mean average precision over all classes. This summarizes the behavior of a method over the whole test set as in the standard VOC protocol [10]; **(DR)** detection rate as the percentage of correctly localized objects over the *positive* test images only. This makes sense as our method returns exactly one window per image (sec. 3.2); **(#win)** the number of window classifier [12] evaluations.

**Baselines and competitors.** We compare our method to two baselines (Sliding Window and Random Chance). Sliding Window is the standard sliding-window scheme of [12], which scores about 25000 windows on a multiscale image grid (for an average VOC image). Random Chance scores 100 randomly sampled windows on the same grid using the same classifier [12].

We also compare to two recent methods designed to reduce the number of classifier evaluations by proposing a limited number of candidate windows likely to cover all objects in the image (Objectness [1] and Selective Search [29])[5]. We apply the detection procedures described in the respective works (sec. 6.1 in [1] and sec. 5.4 in [29]).

**Results.** Table 1 reports performance for all methods we compare. As a reference, Sliding Window [12] reaches a good detection accuracy, but at the price of evaluating many windows (25000). Random Chance fails entirely and achieves a very low detection accuracy, showing that an intelligent search strategy is necessary when evaluating very few windows (100). The two competing methods [1, 29] exhibit a trade-off: they evaluate fewer windows than Sliding Window, but at the price of losing some detection accuracy (confirming what reported in [1, 29]).

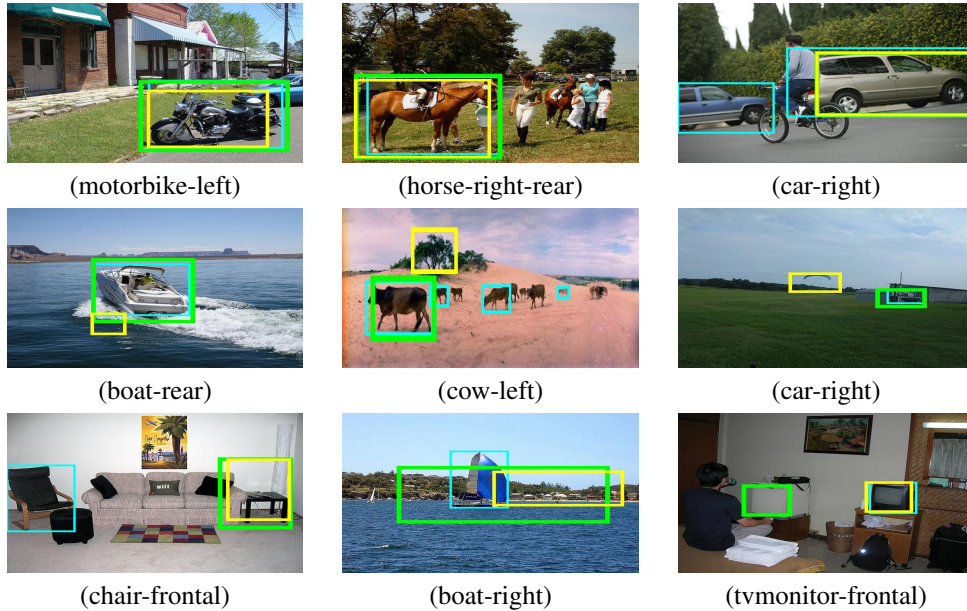

| (motorbike-left) | (horse-right-rear) | (car-right) |
| (boat-rear) | (cow-left) | (car-right) |
| (chair-frontal) | (boat-right) | (tvmonitor-frontal) |

Figure 4: **Qualitative results on PASCAL VOC10.** *Nine example images along with the the ground-truth (cyan), output of our strategy (green) and of Sliding Window (yellow). First row: examples where both methods correctly detect the object. For the car-right example our strategy outputs exactly the same window as Sliding Window. Second row: examples where our method succeeds but Sliding Window fails, because it avoids evaluating cluttered areas where the window classifier [12] produces false positives. Third row: examples where our strategy fails to localize the object. Although when this happens typically Sliding Windows fails too (first two columns), in some rare cases only our strategy fails (third column).*

Our method performs best, as it achieves higher detection performance than Sliding Window (+3.7% DR, +2.7% mAP), while at the same time greatly reducing the number of evaluated windows ($250\times$ fewer). Overall, our method evaluates only as many windows as Random Chance (100). This is $4\times$-$10\times$ fewer than both [1, 29] which were designed for this purpose. The fact that our method achieves higher detection accuracy than Sliding Window might seem surprising at first, as it evaluates a subset of its windows. The reason is that our method exploits context to avoid evaluating large portions of the image, which often contain highly cluttered areas where the window classifier [12] risks producing false-positives (fig. 4).

**Computational efficiency and generality.** While our method greatly reduces the number of windows evaluated, it introduces two overheads: (1) nearest neighbor lookup takes between $2.5s$ and $5.7s$, depending on the class (as the number $L$ of training windows varies, see sec. 4); (2) updating the vote map takes $0.26s$. All timings are totals over the 100 time steps to detect one class in one test image on a single core of a Intel Core i7 3.4GHz CPU. Our total detection time for an average class (5s) is moderately shorter that scoring all windows on the grid (8s), as [12] is already very efficient.

Importantly, our method is general in that it can be applied on top of *any* window classifier. For an expensive classifier, the overhead becomes negligible and we can achieve a substantial speedup. To prove this point we use an intersection kernel SVM [23] on a 3-level spatial pyramid of dense SURF [3] bag-of-words. Note how similar classifiers are used in several object detectors [15, 27, 30]. On an average image containing 25000 windows, sliding window takes 92s to run, whereas our method takes only 8s, hence achieving a $11\times$ speedup (at no loss of mAP).

**Conclusions and future work.** We have proposed a novel object detection technique to replace sliding window with an intelligent search strategy which exploits context to reduce the number of window evaluations and improve detection performance. Our method is general and can be used on top of any window classifier.

As future work we plan to consider richer policy parameterizations and also to improve our learning procedure to optimize the full objective, i.e. the expected detection performance after $T$ time steps, framing it, for instance, as a reinforcement learning problem similar to [5, 25].

**Acknowledgments.** NH and YWT acknowledge funding from the European Community's 7th Framework Programme (FP7/2007-2013) under grant agreement no 270327 and from the Gatsby Charitable foundation.

## Footnotes

[1] In all experiments we use $Z = 10$.

[2]In the experiments we set $\alpha = 0.5$.

[3]In practice we introduce an additional parameter $\beta$ not present in eq. (5) :    we use $\tilde{M}^t(\mathbf{w}|\hat{\mathbf{w}}^1, \hat{\mathbf{y}}^1, \ldots, \hat{\mathbf{w}}^t, \hat{\mathbf{y}}^t; \Theta) = 1/Z(\theta, \hat{\mathbf{h}}, \beta) M^t(\mathbf{w}|\hat{\mathbf{w}}^1, \hat{\mathbf{y}}^1, \ldots, \hat{\mathbf{w}}^t, \hat{\mathbf{y}}^t; \Theta)^\beta$. $\beta$ acts as an inverse temperature and interpolates between a uniform policy (for $\beta \to 0$) and a policy that always selects the highest probability window as in eq. (1) (for $\beta \to \infty$). We use $Z = 100$, $L = 100000$.

[4]The current procedure can be seen as an approximation to stochastic gradient ascent in the full sequential objective $\mathcal{J}(\theta) = \sum_{t=1}^T \sum_{\mathbf{h}^{t-1}} p^M(\mathbf{h}^{t-1}) \left[ \sum_{\mathbf{w}} \mathbf{M}(\mathbf{w}|\mathbf{h}^{t-1}) r(\mathbf{w}) \right]$, where $\mathbf{h}^{t-1} = (\mathbf{w}^1, y^1, \ldots \mathbf{w}^t, \mathbf{y}^t)$, $p^M(\mathbf{h}^t)$ is the distribution over observation sequences of length $t$ resulting from the stochastic policy $M$, and $r(\mathbf{w}) = K_S(\mathbf{w}, \mathbf{w}_{\text{GT}}^b)$. The approximation arises as we ignore changes in the distribution over the observation sequences that are induced by the the changes in parameters, i.e. we assume $\partial p^M / \partial \theta = 0$.

[5]We use the implementation provided by the respective authors, available at `www.vision.ee.ethz.ch/~calvin/objectness/` and `disi.unitn.it/~uijlings/homepage/pmwiki.php?n=Main.Software/`

# References

[1] B. Alexe, T. Deselaers, and V. Ferrari. What is an object? In *CVPR*, 2010.

[2] J. Arpit, R. Saiprasad, and M. Anurag. Multi-stage contour based detection of deformable objects. In *ECCV*, 2008.

[3] H. Bay, A. Ess, T. Tuytelaars, and L. van Gool. SURF: Speeded up robust features. *CVIU*, 110(3):346–359, 2008.

[4] L. Bazzani, N. de Freitas, H. Larochelle, V. Murino, and J. Ting. Learning attentional policies for tracking and recognition in video with deep networks. In *ICML*, 2011.

[5] N. J. Butko and J. R. Movellan. Optimal scanning for faster object detection. In *CVPR*, 2009.

[6] M. Choi, J. Lim, A. Torralba, and A. Willsky. Exploiting hierarchical context on a large database of object categories. In *CVPR*, 2010.

[7] N. Dalal and B Triggs. Histogram of Oriented Gradients for Human Detection. In *CVPR*, 2005.

[8] C Desai, D. Ramanan, and C. Fowlkes. Discriminative models for multi-class object layout. In *ICCV*, 2009.

[9] W. Einhauser and P. Konig. Does luminance-contrast contribute to saliency map for overt visual attention. *European Journal of Neuroscience*, 5(17):1089–1097, 2003.

[10] M. Everingham et al. The PASCAL Visual Object Classes Challenge 2010 Results, 2010.

[11] P. Felzenszwalb, R. Girshick, and D. McAllester. Cascade object detection with deformable part models. In *CVPR*, 2010.

[12] P. Felzenszwalb, R. Girshick, D. McAllester, and D. Ramanan. Object detection with discriminatively trained part based models. *IEEE Trans. on PAMI*, 32(9):1627–1645, 2010.

[13] D. Gao and N. Vasconcelos. Bottom-up saliency is a discriminant process. In *ICCV*, 2007.

[14] Y. Gong and S. Lazebnik. Iterative quantization: A procrustean approach to learning binary codes. In *CVPR*, 2011.

[15] H. Harzallah, F. Jurie, and C. Schmid. Combining efficient object localization and image classification. In *ICCV*, 2009.

[16] G. Heitz and D. Koller. Learning spatial context: Using stuff to find things. In *ECCV*, 2008.

[17] X. Hou and L. Zhang. Saliency detection: A spectral residual approach. In *CVPR*, 2007.

[18] L. Itti, C. Koch, and E. Niebur. A model of saliency-based visual attention for rapid scene analysis. *IEEE Trans. on PAMI*, 20(11):1254–1259, 1998.

[19] G. Krieger, I. Rentschler, G. Hauske, K. Schill, and C. Zetzsche. Object and scene analysis by saccadic eye-movements: an investigation with higher-order statistics. *Spatial Vision*, 2(16):201–214, 2000.

[20] C. H. Lampert, M. B. Blaschko, and T. Hofmann. Beyond sliding windows: Object localization by efficient subwindow search. In *CVPR*, 2008.

[21] H. Larochelle and G. E. Hinton. Learning to combine foveal glimpses with a third-order Boltzmann machine. In *NIPS*, 2010.

[22] B. Leibe, A. Leonardis, and B. Schiele. Combined object categorization and segmentation with an implicit shape model. In *Workshop on Statistical Learning in Computer Vision, ECCV*, May 2004.

[23] S. Maji, A. Berg, and J. Malik. Classification using intersection kernel support vector machines is efficient. In *CVPR*, 2008.

[24] J. Najemnik and W. S. Geisler. Optimal eye movement strategies in visual search. *Nature*, 434:381–391, 2005.

[25] L. Paletta, G. Fritz, and C. Seifert. Q-learning of sequential attention for visual object recognition from informative local descriptors. In *ICML*, 2005.

[26] M. Pedersoli, A. Vedaldi, and J. Gonzales. A coarse-to-fine approach for fast deformable object detection. In *CVPR*, 2011.

[27] A. Prest, C. Leistner, J. Civera, C. Schmid, and V. Ferrari. Learning object class detectors from weakly annotated video. In *CVPR*, 2012.

[28] A. Rabinovich, A. Vedaldi, C. Galleguillos, E. Wiewiora, and S. Belongie. Objects in context. In *ICCV*, 2007.

[29] K. Van de Sande, J. Uijlings, T. Gevers, and A. Smeulders. Segmentation as selective search for object recognition. In *ICCV*, 2011.

[30] A. Vedaldi, V. Gulshan, M. Varma, and A. Zisserman. Multiple kernels for object detection. In *ICCV*, 2009.

[31] P. Viola and M. Jones. Rapid object detection using a boosted cascade of simple features. In *CVPR*, 2001.

